# Factorizing Multivariate Function Classes

**Juan K. Lin**[*]
Department of Physics
University of Chicago
Chicago, IL 60637

## Abstract

The mathematical framework for factorizing equivalence classes of multivariate functions is formulated in this paper. Independent component analysis is shown to be a special case of this decomposition. Using only the local geometric structure of a class representative, we derive an analytic solution for the factorization. We demonstrate the factorization solution with numerical experiments and present a preliminary tie to decorrelation.

## 1 FORMALISM

In independent component analysis (ICA), the goal is to find an unknown linear coordinate system where the joint distribution function admits a factorization into the product of one dimensional functions. However, this decomposition is only rarely possible. To formalize the notion of multivariate function factorization, we begin by defining an equivalence relation.

**Definition.** We say that two functions $f, g : \mathbb{R}^n \to \mathbb{R}$ are *equivalent* if there exists $A, \vec{b}$ and $c$ such that: $f(\vec{x}) = cg(A\vec{x} + \vec{b})$, where $A$ is a non-singular matrix and $c \neq 0$.

Thus, the equivalence class of a function consists of all invertible linear transformations of it. To avoid confusion, equivalence classes will be denoted in upper case, and class representatives in lower case. We now define the *product* of two equivalence classes. Consider representatives $b : \mathbb{R}^n \to \mathbb{R}$, and $c : \mathbb{R}^m \to \mathbb{R}$ of corresponding equivalence classes $B$ and $C$. Let $\vec{x_1} \in \mathbb{R}^n$, $\vec{x_2} \in \mathbb{R}^m$, and $\vec{x} = (\vec{x_1}, \vec{x_2})$. From the scalar product of the two functions, define the function $a : \mathbb{R}^{n+m} \to \mathbb{R}$ by $a(\vec{x}) = b(\vec{x_1})c(\vec{x_2})$. Let the product of $B$ and $C$ be the equivalence class $A$ with

---

[*] Current address: E25-201, MIT, Cambridge, MA 02139. Email: jklin@ai.mit.edu

representative $a(\vec{x})$. This product is independent of the choice of representatives of $B$ and $C$, and hence is a well defined operation on equivalence classes. We proceed to define the notion of an *irreducible* class.

**Definition.** Denote the equivalence class of constants by $I$. We say that $A$ is *irreducible* if $A = BC$ implies either $B = A, C = I$, or $B = I, C = A$.

From the way products of equivalence classes are defined, we know that all equivalence classes of one dimensional functions are irreducible. Our formulation of the factorization of multivariate function classes is now complete. Given a multivariate function, we seek a factorization of the equivalence class of the given representative into a product of irreducibles. Intuitively, in the context of joint distribution functions, the irreducible classes constitute the underlying sources. This factorization generalizes independent component analysis to allow for higher dimensional "vector" sources. Consequently, this decomposition is well–defined for *all* multivariate function classes. We now present a local geometric approach to accomplishing this factorization.

## 2   LOCAL GEOMETRIC INFORMATION

Given that the joint distribution factorizes into a product in the "source" coordinate system, what information can be extracted locally from the joint distribution in a "mixed" coordinate frame? We assume that the relevant multivariate function is twice differentiable in the region of interest, and denote $H^f$, the Hessian of $f$, to be the matrix with elements $H^f_{ij} = \partial_i \partial_j f$, where $\partial_k = \frac{\partial}{\partial s_k}$.

**Proposition:** $H^f$ is block diagonal everywhere, $\partial_i \partial_j f|_{\vec{s_0}} = 0$ for all points $\vec{s_0}$ and all $i \le k$, $j > k$, *if and only if* $f$ is separable into a sum $f(s_1, \ldots, s_n) = g(s_1, \ldots, s_k) + h(s_{k+1}, \ldots, s_n)$ for some functions $g$ and $h$.

**Proof – Sufficiency:**
Given $f(s_1, \ldots, s_n) = g(s_1, \ldots, s_k) + h(s_{k+1}, \ldots, s_n)$,

$$\frac{\partial^2 f}{\partial s_i \partial s_j} = \frac{\partial}{\partial s_i} \frac{\partial h(s_{k+1}, \ldots, s_n)}{\partial s_j} = 0$$

everywhere for all $i \le k$, $j > k$.

**Necessity:**
From $H^f_{1n} = 0$, we can decompose $f$ into

$$f(s_1, s_2, \ldots, s_n) = \hat{g}(s_1, \ldots, s_{n-1}) + \hat{h}(s_2, \ldots, s_n),$$

for some functions $\hat{g}$ and $\hat{h}$. Continuing by imposing the constraints $H^f_{1j} = 0$ for all $j > k$, we find

$$f(s_1, s_2, \ldots, s_n) = \breve{g}(s_1, \ldots, s_k) + \breve{h}(s_2, \ldots, s_n).$$

Combining with $H^f_{2j} = 0$ for all $j > k$ yields

$$f(s_1, s_2, \ldots, s_n) = \tilde{g}(s_1, \ldots, s_k) + \tilde{h}(s_3, \ldots, s_n).$$

Finally, inducting on $i$, from the constraints $H^f_{ij} = 0$ for all $i \le k$ and $j > k$, we arrive at the desired functional form

$$f(s_1, s_2, \ldots, s_n) = g(s_1, \ldots, s_k) + h(s_{k+1}, \ldots, s_n).$$

More explicitly, a twice–differentiable function satisfies the set of coupled partial differential equations represented by the block diagonal structure of $H$ if and only if it admits the corresponding separation of variables decomposition. By letting $\log p = f$, the additive decomposition of $f$ translates to a product decomposition of $p$. The more general decomposition into an arbitrary number of factors is obtained by iterative application of the above proposition. The special case of independent component analysis corresponds to a strict diagonalization of $H$. Thus, in the context of smooth joint distribution functions, pairwise conditional independence is necessary and sufficient for statistical independence.

To use this information in a transformed "mixture" frame, we must understand how the matrix $H^{\log p}$ transforms. From the relation between the mixture and source coordinate systems given by $\vec{x} = A\vec{s}$, we have $\frac{\partial}{\partial s_i} = A_{ji}\frac{\partial}{\partial x_j}$, where we use Einstein's convention of summation over repeated indices. From the relation between the joint distributions in the mixture and source frames, $p_s(\vec{s}) = |A|p_x(\vec{x})$, direct differentiation gives

$$\frac{\partial^2 \log p_s(\vec{s})}{\partial s_i \partial s_l} = A_{ji} A_{kl} \frac{\partial^2 \log p_x(\vec{x})}{\partial x_j \partial x_k}.$$

Letting $H_{ij} = \frac{\partial^2 \log p_s(\vec{s})}{\partial s_i \partial s_j}$ and $\tilde{H}_{ij} = \frac{\partial^2 \log p_x(\vec{x})}{\partial x_i \partial x_j}$, in matrix notation we have $H = A^T \tilde{H} A$. In other words, $H$ is a second rank (symmetric) covariant tensor. The joint distribution admits a product decomposition in the source frame if and only if $H$ and hence $A^T \tilde{H} A$ has the corresponding block diagonal structure. Thus multivariate function class factorization is solved by joint block diagonalization of symmetric matrices, with constraints on $A$ of the form $A_{ji}\tilde{H}_{jk}A_{kl} = 0$.

Because the Hessian is symmetric, its diagonalization involves only ($n$ choose 2) constraints. Consequently, in the independent component analysis case where the joint distribution function admits a factorization into one dimensional functions, if the mixing transformation is orthogonal, the independent component coordinate system will lie along the eigenvector directions of $\tilde{H}$. Generally however, $n(n-1)$ independent constraints corresponding to information from the Hessian at two points are needed to determine the $n$ arbitrary coordinate directions.

## 3  NUMERICAL EXPERIMENTS

In the simplest attack on the factorization problem, we solve the constraint equations from two points simultaneously. The analytic solution is demonstrated in two dimensions. Without loss of generality, the mixing matrix $A$ is taken to be of the form

$$A = \begin{pmatrix} 1 & x \\ y & 1 \end{pmatrix}.$$

The constraints from the two points are: $ax + b(xy + 1) + cy = 0$, and $a'x + b'(xy + 1) + c'y = 0$, where $H_{11} = a$, $H_{21} = H_{12} = b$ and $H_{22} = c$ at the first point, and the primed coefficients denote the values at the second point.

Solving the simultaneous quadratic equations, we find

$$x = \frac{a'c - ac' \pm \sqrt{(a'c - ac')^2 - 4(a'b - ab')(b'c - bc')}}{2(ab' - a'b)},$$

$$y = \frac{a'c - ac' \pm \sqrt{(a'c - ac')^2 - 4(a'b - ab')(b'c - bc')}}{2(bc' - b'c)}.$$

The $\pm$ double roots is indicative of the $(x,y) \to (1/y, 1/x)$ symmetry in the equations, and together only give two distinct orientation solutions. These independent component orientation solutions are given by $\theta_1 = \tan^{-1}(1/x)$ and $\theta_2 = \tan^{-1}(y)$.

## 3.1  Natural Audio Sources

To demonstrate the analytic factorization solution, we present some proof of concept numerics. Generality is pursued over optimization concerns. First, we perform the standard separation of two linearly mixed natural audio sources. The input dataset consists of 32000 un-ordered datapoints, since no use will be made of the temporal information. The process for obtaining estimates of the Hessian matrix $\tilde{H}$ is as follows. A histogram of the input distribution was first acquired and smoothed by a low-pass Gaussian mask in spatial-frequency space. The elements of $\tilde{H}$ were then obtained via convolution with a discrete approximation of the derivative operator. The width of the Gaussian mask and the support of the derivative operator were chosen to reduce sensitivity to low spatial-frequency uncertainty. It should be noted that the analytic factorization solution makes no assumptions about the mixing transformation, consequently, a blind determination of the smoothing length scale is not possible because of the multiplicative degree of freedom in each source.

Because of the need to take the logarithm of $p$ before differentiation, or equivalently to divide by $p$ afterwards, we set a threshold and only extracted information from points where the number of counts was greater than threshold. This is justified from a counting uncertainty perspective, and also from the understanding that regions with vanishing probability measure contain no information.

With our sample of 32000 datapoints, we considered only the bin-points with a corresponding bin count greater than 30. From the 394 bin locations that satisfied this constraint, the solutions $(\theta_1, \theta_2)$ for all (394 choose 2) $= (394 \cdot 393/2)$ pairs of the corresponding factorization equations are plotted in Fig. 1. A histogram of these solutions are shown in Fig. 2. The two peaks in the solution histogram correspond to orientations that differ from the two actual independent component orientations by 0.008 and 0.013 radians. The signal to mixture ratio of the two outputs generated from the solution are 158 and 49.

## 3.2  Effect of Noise

Because the solution is analytic, uncertainty in the sampling just propagates through to the solution, giving rise to a finite width in the solution's distribution. We investigated the effect of noise and counting uncertainty by performing numerics starting from analytic forms for the source distributions. The joint distribution in the source frame was taken to be:

$$p_s(s_1, s_2) = (2 + \sin(s_1)) * (2 + \sin(s_2)).$$

Normalization is irrelevant since a function's decomposition into product form is preserved in scalar multiplication. This is also reflected in the equivalence between $H^{\log p}$ and $H^{\log cp}$ for $c$ an arbitrary positive constant. The joint distribution in the mixture frame was obtained from the relation $p_x(\vec{x}) = |A|^{-1} p_s(\vec{s})$. To simulate

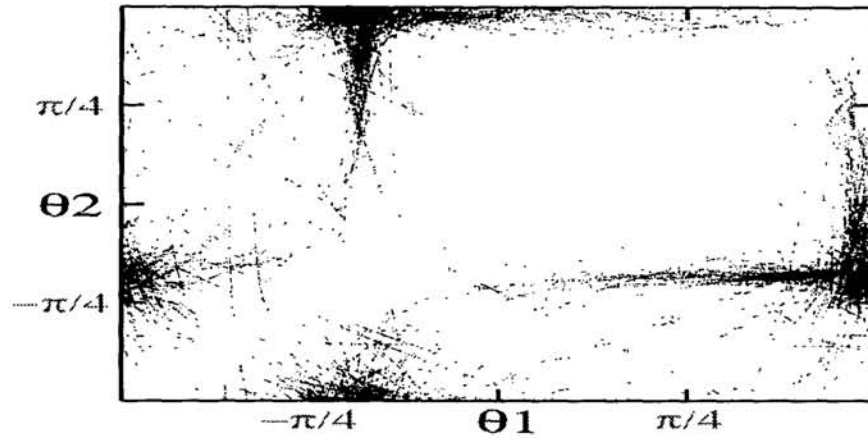

Figure 1: Scatterplot of the independent component orientation solutions. All unordered solution pairs $(\theta_1, \theta_2)$ are plotted. The solutions are taken in the range from $-\pi/2$ to $\pi/2$.

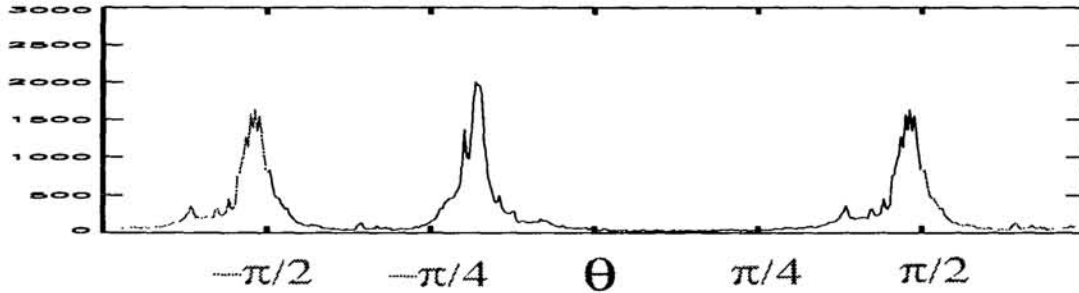

Figure 2: Histogram of the orientation solutions plotted in the previous figure. The range is still taken from $-\pi/2$ to $\pi/2$, with the histogram wrapped around to ease the circular identification. The mixing matrix used was: $a_{11} = 0.0514$, $a_{21} = 0.779$, $a_{12} = 0.930$, $a_{22} = -0.579$, giving independent component orientations at $-0.557$ and $1.505$ radians. Gaussian fit to the centers of the two solution peaks give $-0.570 \pm 0.066$ and $1.513 \pm 0.077$ radians for the two orientations.

sampling, $p_x(\vec{x})$ was multiplied with the number of samples $M$, onto which was added Gaussian distributed noise with amplitude given by the $(M\, p_x(\vec{x}))^{1/2}$. This reflects the fact that counting uncertainty scales as the square root of the number of counts. The result was rounded to the nearest integer, with all negative count values set to zero. The subsequent processing coincided with that for natural audio sources. From the source distribution equation above, the minimum number of expected counts is $M$, and the maximum is $9M$. The results in Figures 3 and 4 show that, as expected, increasing the number of samplings decreases the widths of the solution peaks. By fitting Gaussians to the two peaks, we find that the uncertainty (peak widths) in the independent component orientations changes from 0.06 to 0.1 radians as the sampling is decreased from for $M = 20$ to $M = 2$. So even with few samplings, a relatively accurate determination of the independent component coordinate system can be made.

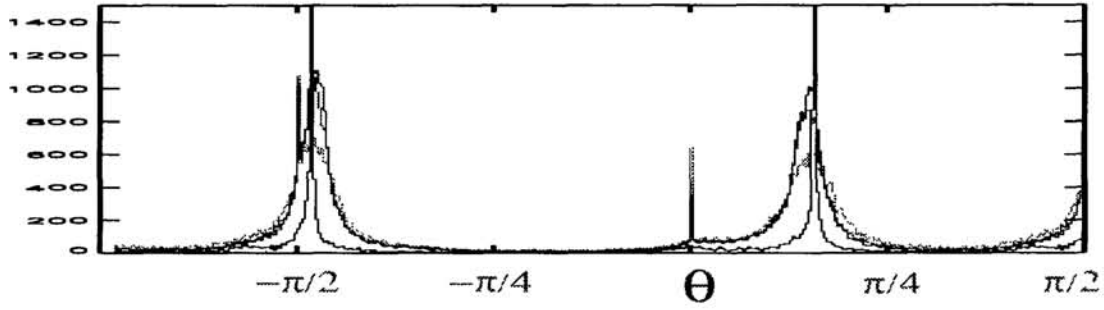

Figure 3: Histogram of the independent component orientation solutions for four different samplings. Solutions were generated from 20000 randomly chosen pairs of positions. The curves, from darkest to lightest, correspond to solutions for the noiseless, $M = 20, 11$ and 2 simulations. The noiseless solution histogram curve extends to a height of approximately 15000 counts, and is accurate to the width of the bin. The slight scatter is due to discretization noise. Spikes at $\theta = 0$ and $-\pi/2$ correspond to pairs of positions which contain no information.

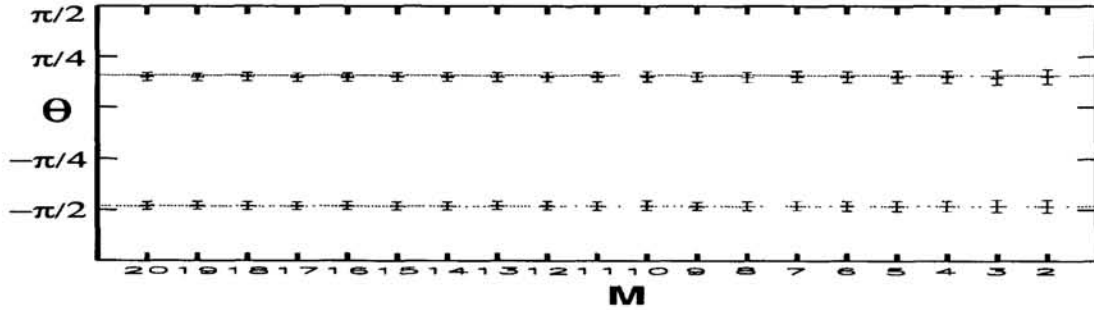

Figure 4: The centers and widths of the solution peaks as a function of the minimum expected number of counts $M$. From the source distribution, the maximum expected number of counts is $9M$. Information was only extracted from regions with more than $2M$ counts. The actual independent component orientation as determined from the mixing matrix $A$ are shown by the two dashed lines. The solutions are very accurate even for small samplings.

## 4   RELATION TO DECORRELATION

Ideally, if a mixed tensor (transforms as $J = A^{-1} \tilde{J} A$) with the full degrees of freedom can be found which is diagonal if and only if the joint distribution appears in product form, then the independent component coordinate directions will coincide with that of the tensor's eigenvectors. However, the preceding analysis shows that a maximum of $n(n-1)/2$ constraints contain all the information that exists locally. This, however, provides a nice connection with decorrelation.

Starting with the characteristic function of $\log p(\vec{x})$, $\phi(\vec{k}) = \int e^{i\vec{k}\cdot\vec{x}} \log p(\vec{x}) \, d\vec{x}$, the off diagonal terms of $H^{\log p}$ are given by

$$\frac{\partial^2 \log p}{\partial x_i \partial x_j} \propto \int k_i k_j e^{-i\vec{k}\cdot\vec{x}} \phi(\vec{k}) \, d\vec{k},$$

which can loosely be seen as the second order cross-moments in $\phi(\vec{k})$. Thus di-

agonalization of $H^{\log p}$ roughly translates into decorrelation in $\phi(\vec{k})$. It should be noted that $\phi(\vec{k})$ is not a proper distribution function. In fact, it is a complex valued function with $\phi(\vec{k}) = \phi^*(-\vec{k})$. Consequently, the summation in the above equation is not an expectation value, and needs to be interpreted as a superposition of plane waves with specified wavelengths, amplitudes and phases.

## 5  DISCUSSION

The introduced functional decomposition defines a generalization of independent component analysis which is valid for all multivariate functions. A rigorous notion of the decomposition of a multivariate function into a set of lower dimensional factors is presented. With only the assumption of local twice differentiability, we derive an analytic solution for this factorization [1]. A new algorithm is presented, which in contrast to iterative non–local parametric density estimation ICA algorithms [2, 3, 4], performs the decomposition analytically using local geometric information. The analytic nature of this approach allows for a proper treatment of source separation in the presence of uncertainty, while the local nature allows for a local determination of the source coordinate system. This leaves open the possibility of describing a position dependent independent component coordinate system with local linear coordinates patches.

The presented class factorization formalism removes the decomposition assumptions needed for independent component analysis, and reinforces the well known fact that sources are recoverable only up to linear transformation. By modifying the equivalence class relation, a rich underlying algebraic structure with both multiplication and addition can be constructed. Also, it is clear that the matrix of second derivatives reveals an even more general combinatorial undirected graphical structure of the multivariate function. These topics, as well as uniqueness issues of the factorization will be addressed elsewhere [5].

The author is grateful to Jack Cowan, David Grier and Robert Wald for many invaluable discussions.

## References

[1] J. K. Lin, *Local Independent Component Analysis*, Ph. D. thesis, University of Chicago, 1997.

[2] A. J. Bell and T. J. Sejnowski, Neural Computation **7**, 1129 (1995).

[3] S. Amari, A. Cichocki, and H. Yang, in *Advances in Neural and Information Processing Systems, 8*, edited by D. S. Touretzky, M. C. Mozer, and M. E. Hasselmo (MIT Press, Cambridge, MA, 1996), pp. 757–763.

[4] B. A. Pearlmutter and L. Parra, in *Advances in Neural and Information Processing Systems, 9*, edited by M. C. Mozer, M. I. Jordan, and T. Petsche (MIT Press, Cambridge, MA, 1997), pp. 613–619.

[5] J. K. Lin, *Graphical Structure of Multivariate Functions*, in preparation.
